# Computing with Action Potentials

**John J. Hopfield**[*]        Carlos D. Brody [†]        Sam Roweis [†]

## Abstract

Most computational engineering based loosely on biology uses continuous variables to represent neural activity. Yet most neurons communicate with action potentials. The engineering view is equivalent to using a rate-code for representing information and for computing. An increasing number of examples are being discovered in which biology may not be using rate codes. Information can be represented using the *timing* of action potentials, and efficiently computed with in this representation. The "analog match" problem of odour identification is a simple problem which can be efficiently solved using action potential timing and an underlying rhythm. By using adapting units to effect a fundamental change of representation of a problem, we map the recognition of words (having uniform time-warp) in connected speech into the same analog match problem. We describe the architecture and preliminary results of such a recognition system. Using the fast events of biology in conjunction with an underlying rhythm is one way to overcome the limits of an event-driven view of computation. When the intrinsic hardware is much faster than the time scale of change of inputs, this approach can greatly increase the effective computation per unit time on a given quantity of hardware.

## 1 Spike timing

Most neurons communicate using action potentials – stereotyped pulses of activity that are propagated along axons without change of shape over long distances by active regenerative processes. They provide a pulse-coded way of sending information. Individual action potentials last about 2 ms. Typical active nerve cells generate 5–100 action potentials/sec.

Most biologically inspired engineering of neural networks represent the activity of a nerve cell by a continuous variable which can be interpreted as the short-time average rate of generating action potentials. Most traditional discussions by neurobiologists concerning how information is represented and processed in the brain have similarly relied on using "short term mean firing rate" as the carrier of information and the basis for computation. But this is often an ineffective way to compute and represent information in neurobiology.

---

[*]Dept. of Molecular Biology, Princeton University. jhopfield@watson.princeton.edu
[†] Computation & Neural Systems, California Institute of Technology.

To define "short term mean firing rate" with reasonable accuracy, it is necessary to either wait for several action potentials to arrive from a single neuron, or to average over many roughly equivalent cells. One of these necessitates slow processing; the other requires redundant "wetware".

Since action potentials are short events with sharp rise times, action potential timing is another way that information can be represented and computed with ([Hopfield, 1995]). Action potential timing seems to be the basis for some neural computations, such as the determination of a sharp response time to an ultrasonic pulse generated by the moustache bat. In this system, the bat generates a 10 ms pulse during which the frequency changes monotonically with time (a "chirp"). In the cochlea and cochlear nucleus, cells which are responsive to different frequencies will be sequentially driven, each producing zero or one action potentials during the time when the frequency is in their responsive band. These action potentials converge onto a target cell. However, while the times of initiation of the action potentials from the different frequency bands are different, the length and propagation speed of the various axons have been coordinated to result in all the action potentials arriving at the target cell at the same time, thus recognizing the "chirped" pulse as a whole, while discriminating against random sounds of the same overall duration.

Taking this hint from biology, we next investigate the use of action potential timing to represent information and compute with in one of the fundamental computational problems relevant to olfaction, noting why the elementary "neural net" engineering solution is poor, and showing why computing with action potentials lacks the deficiencies of the conventional elementary solution.

## 2  Analog match

The simplest computational problem of odors is merely to identify a known odor when a single odor dominates the olfactory scene. Most natural odors consist of mixtures of several molecular species. At some particular strength a complex odor $b$ can be described by the concentrations $N_i^b$ of its constitutive molecular of species $i$. If the stimulus intensity changes, each component increases (or decreases) by the same multiplicative factor. It is convenient to describe the stimulus as a product of two factors, an intensity $\lambda$ and normalized components $n_i^b$ as:

$$\lambda = \Sigma_j N_j^b \quad \Rightarrow \quad n_i^b = N_i^b / \lambda \quad \text{or} \quad N_i^b = \lambda n_i^b \tag{1}$$

The $n_i^b$ are normalized, or relative concentrations of different molecules, and $\lambda$ describes the overall odor intensity. Ideally, a given odor quality is described by the pattern of $n_i^b$, which does not change when the odor intensity $\lambda$ changes. When a stimulus $s$ described by a set $\{N_j^s\}$ is presented, an ideal odor quality detector answers "yes" to the question "is odor $b$ present?" if and only if for some value of $\lambda$:

$$N_j^s \approx \lambda n_j^b \quad \forall j \tag{2}$$

This general computation has been called *analog match*.[1]

The elementary "neural net" way to solve analog match and recognize a single odor independent of intensity would be to use a single "grandmother unit" of the following type.

Call the unknown odor vector $I$, and the weight vector $W$. The input to the unit will then be $I \cdot W$. If $W = n/\|n\|$ and $I$ is pre-normalized by dividing by the Euclidean magnitude $\|I\|$, recognition can be identified by $I \cdot W > .95$, or whatever threshold describes the degree of precision in identification which the task requires.

This solution has four major weaknesses.

1. Euclidean normalization is used; not a trivial calculation for real neural hardware.

2. The size of input components $I_k$ and their importance is confounded. If a weak component has particular importance, or a strong one is not reliable, there is no way to represent this. $W$ describes only the size of the target odor components.

3. There is no natural composition if the problem is to be broken into a hierarchy by breaking the inputs into several parts, solving independently, and feeding these results on to a higher level unit for a final recognition. This is best seen by analogy to vision. If I recognize in a picture grandmother's nose at one scale, her mouth at another, and her right eye at a third scale, then it is assuredly *not* grandmother. Separate normalization is a disaster for creating hierarchies.

4. A substantial number of inputs may be missing or giving grossly wrong information. The "dot-product-and-threshold" solution cannot contend with this problem. For example, in olfaction, two of the common sources of noise are the adaptation of a subset of sensors due to previous strong odors, and receptors stuck "on" due to the retention of strongly bound molecules from previous odors.

All four problems are removed when the information is encoded and computed with in an action potential representation, as illustrated below. The three channels of analog input $I_a, I_b, I_c$ are illustrated on the left. They are converted to a spike timing representation by the position of action potentials with respect to a fiducial time $T$. The interval between $T$ and the time of an action potential in a channel $j$ is equal to $\log I_j$. Each channel is connected to an output unit through a delay line of length $\Delta_j = \log n_j^b$, where $n^b$ is the target vector to be identified. When the analog match criterion is satisfied, the pulses on all three channels will arrive at the target unit at the same time, driving it strongly. If all inputs are scaled by $\alpha$, then the times of the action potentials will all be changed by $\log \alpha$. The three action potentials will arrive at the recognition unit simultaneously, but a a time shifted by $\log \alpha$. Thus a pattern can be recognized (or not) on the basis of its relative components. Scale information is retained in the *time* at which the recognition unit is driven. The system clearly "composes", and difficulty (3) is surmounted. No normalization is required, eliminating difficulty (1). Each pathway has two parameters describing it, a delay (which contains the information about the pattern to be recognized) and a synaptic strength (which describes the weight of the action potential at the recognition unit). Scale and importance are separately represented. The central computational motif is very similar to that used in bat sonar, using relative timing to represent information and time delays to represent target patterns.

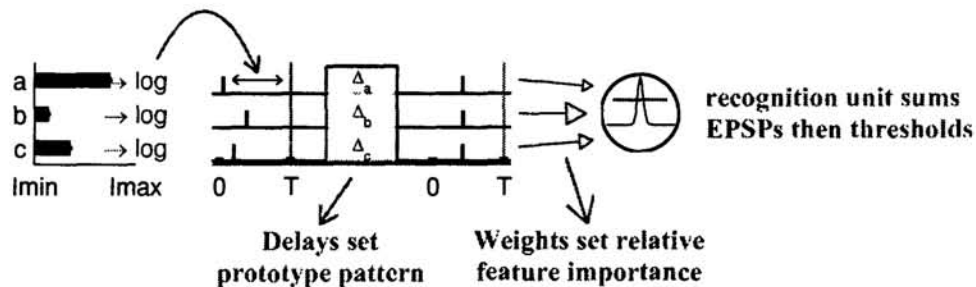

Delays set                  Weights set relative
prototype pattern          feature importance

This system also tolerates errors due to missing or grossly inaccurate information. The figure below illustrates this fact for the case of three inputs, and contrasts the receptive fields of a system computing with action potentials with those of a conventional grandmother cell. (The only relevant variables are the projections of the input vector on the surface of the unit sphere, as illustrated.) When the thresholds are set high, both schemes recognize a small, roughly circular region around the target pattern (here chosen as 111). Lowering the recognition threshold in the action-potential based scheme results in a star-shaped region being recognized; this region can be characterized as "recognize if any two components are in the correct ratio, independent of the size of the third component." Pattern 110 is thus recognized as being similar to 111 while still rejecting most of the space as not resembling the target. In contrast, to recognize 110 with the conventional unit requires such threshold lowering that almost any vector would be recognized.

### Spike Timing                    Normalize, Dot Product

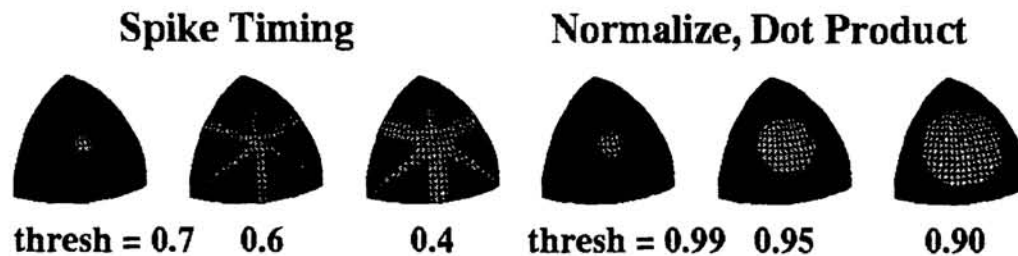

thresh = 0.7    0.6        0.4        thresh = 0.99   0.95        0.90

This method of representation and computation using action potential timing requires a fiducial time available to all neurons *participating in stimulus encoding*. Fiducial times might be externally generated by salient events, as they are in the case of moustache bat sonar. Or they could be internally generated, sporadically or periodically. In the case of the olfactory system, the first processing area of all animals has an oscillatory behavior. A large piece of the biophysics of neurons can be represented by the idea that neurons are leaky integrators, and that when their internal potential is pushed above a critical value, they produce an action potential, and their internal potential is reset a fixed distance below threshold. When a sub-threshold input having a frequency $f$ is combined with a steady analog current $I$, the system generates action potentials at frequency $f$, but whose phase with respect to the underlying oscillation is a monotone function of $I$. Thus the system encodes $I$ into a phase (or time) of an action potential with respect to the underlying rhythm. Interestingly, in mammals, the second stage of the olfactory system, the prepiriform cortex, has slow axons propagating signals across it. The propagation time delays are comparable to $1/f$. The system has the capability of encoding and analyzing information in action potential timing.

## 3    Time warp and speech

Recognizing syllables or words independent of a uniform stretch ("uniform time warp") can in principle be cast as an analog match problem and transformed into neural variables [Hopfield, 1996]. We next describe this approach in relationship to a previous "neural network" way of recognizing words in connected speech [Hopfield and Tank, 1987, Unnikrishnan et al., 1991, Unnikrishnan et al., 1992] (UHT for short).

A block diagram below shows the UHT neural network for recognizing a small vocabulary of words in connected speech. The speech signal is passed through a bank of band-pass filters, and an elementary neural feature detector then examines whether each frequency is a local maximum of the short-term power spectrum. If so, it propagates a "1" down a delay line from that feature detector, thus converting the pattern of features in time into a pattern in space. The recognition unit for a particular word is then connected to these delay lines by a pattern of weights which are trained on a large data base.

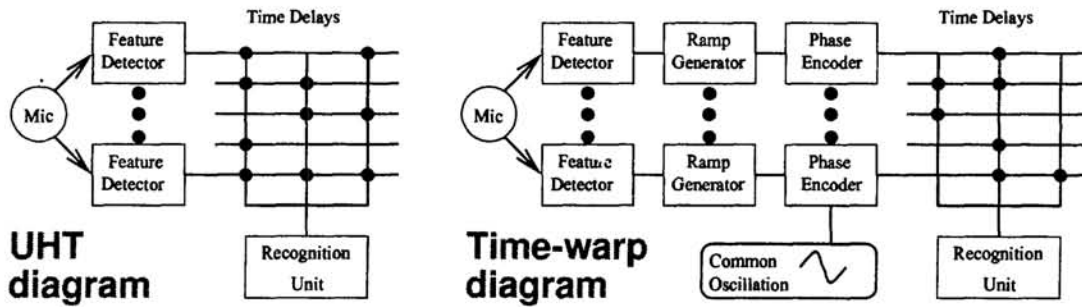

**UHT diagram**

**Time-warp diagram**

The conceptual strength of this circuit is that it requires no indication of the boundaries between words. Indeed, there is no such concept in the circuit. The conceptual weakness of this "neural network" is that the recognition process for a particular word is equivalent to sliding a rigid template across the feature pattern. Unfortunately, even a single speaker has great variation in the duration of a given word under different circumstances, as illustrated in the two spectrograms below. Clearly no single template will fit these both of these utterances of "one" very well. This general problem is known as *time-warp*. A time-warp invariant recognizer would have considerable advantage.

**Two instances of "one"**

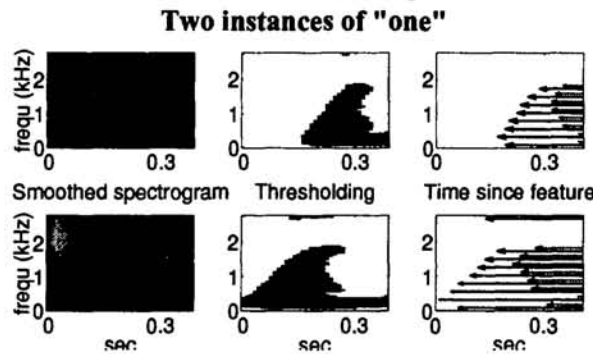

Smoothed spectrogram     Thresholding     Time since feature

The UHT approach represents a sequence by the presence of a signal on feature signal lines A, B, C, as shown on the left of the figure below. Suppose the end of the word occurs at some particular time as indicated. Then the feature starts and stops can be described as an analog vector of times, whose components are shown by the arrows as indicated. In this representation, a word which is spoken more slowly simply has all its vector components multiplied by a common factor. The problem of recognizing words within a uniform time warp is thus isomorphic with the analog match problem, and can be readily solved by using action potential timing and an underlying rhythm, as described above. In our present modeling, the rhythm has a frequency of 50 Hz, significantly faster than the rate at which new features appear in speech. This frequency corresponds to the clock rate at which speech features are effectively "sampled". In the UHT circuit this rate was set by the response timescale of the recognition units. But where each template in the UHT circuit attempted only a single match with the feature vector per sample, this circuit allows the attempted match of many possible time-warps with the feature vector per sample. (The range of time-warps allowed is determined by the oscillation frequency and the temporal resolution of the spike timing system.)

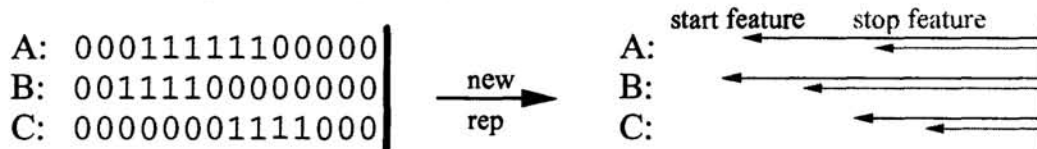

A: 00011111100000
B: 00111100000000
C: 00000001111000

The block diagram of the neural circuit necessary to recognize words in connected speech with uniform time warp is sketched above. It looks superficially similar to the UHT circuit beside it, except for the insertion of a ramp generator and a phase encoder between the

feature detectors and the delay system. Recognizing a feature activates a ramp generator whose output decays. This becomes the input to a "neuron" which has an additional oscillatory input at frequency $f$. If the ramp decay and oscillation shapes are properly matched, the logarithm of the time since the occurrence of a feature is encoded in action potential timing as above. Following this encoding system there is a set of tapped delay lines of the same style which would have been necessary to solve the olfactory decoding problem. The total the amount of hardware is similar to the UHT approach because the connections and delay lines dominate the resource requirements.

The operation of the present circuit is, however, entirely different. What the present circuit does is to "remember" recent features by using ramp generators, encode the logarithms of times since features into action potential timing, and recognize the pattern with a time-delay circuit. The time delays in the present circuit have an entirely different meaning from those of the UHT circuit, since they are dimensionally not physical time, but instead are a representation of the logarithm of feature times. The time delays are only on the scale of $1/f$ rather than the duration of a word. There are simple biological implementations of these ideas. For example, when a neuron responds, as many do, to a step in its input by generating a train of action potentials with gradually falling firing frequency (adaptation), the temporal spacing between the action potentials is an implicit representation of the time since the "step" occurred (see [Hopfield, 1996]).

For our initial engineering investigations, we used very simple features. The power within each frequency band is merely thresholded. An upward crossing of that threshold represents a "start" feature for that band, and a downward crossing a "end" feature. A pattern of such features is identified above beside the spectrograms. Although the pattern of feature vectors for the two examples of "one" do not match well because of time warp, when the logarithms of the patterns are taken, the difference between the two patterns is chiefly a shift, i.e. the dominant difference between the patterns is merely uniform time warp.

To recognize the spoken digit "one", for example, the appropriate delay for each channel was chosen so as to minimize the variance of the post-delay spike times (thus aligning the spikes produced by all features), averaged over the different exemplars which contained that feature. All channels with a feature present were given a unity weight connection at that delay value; inactive channels were given weight zero. The figure below shows, on the left, the spike input to the recognition unit (top) and the sum of the EPSPs caused by these inputs (bottom). The examples of "one" produced maximum outputs in different cycles of the oscillation, corresponding to the actual "end times" at which the words should be viewed as recognized. Only the maximum cycle for each utterance is shown here. Within their maximum cycle, different examples of the utterances produced maximal outputs at different phases of the cycle, corresponding to the fact that the different utterances were recognized as having different time warp factors. The panels on the right show the result of playing spoken "four"s into the same recognition unit.

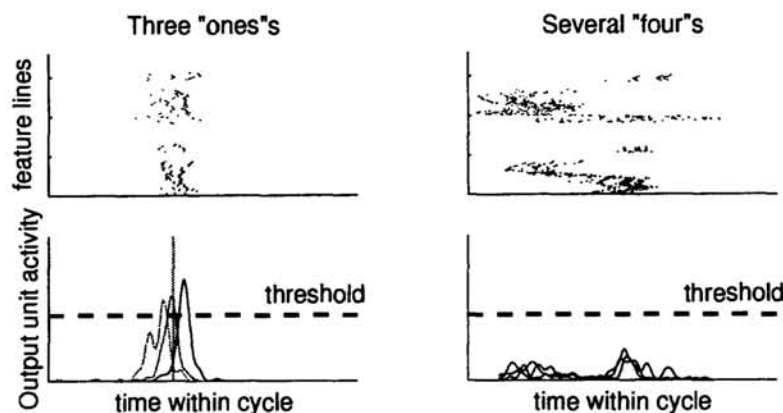

There is no difficulty in distinguishing "ones" from other digits. When, however, the possibility of adjusting the time-warp is turned off, resulting in a "rigid" template it was not possible to discriminate between "one" and other digits. (Disabling time-warp effectively forces recognition to take place at the same "time" during each oscillation. Imagine drawing a vertical line in the figure and notice that it cannot pass through all the peaks of output unit activities.)

We have described the beginning of a research project to use action potentials and timing to solve a real speech problem in a "neural" fashion. Very unsophisticated features were used, and no competitive learning was employed in setting the connection weights. Even so, the system appears to function in a word-spotting mode, and displays a facility of matching patterns with time warp. Its intrinsic design makes it insensitive to burst noise and to frequency-band noise.

How is computation being done? After features are detected, rates of change are slow, and little additional information is accumulated during say a 50 ms. interval. If we let "time be its own representation", as Carver Mead used to say, we let the information source be the effective clock, and the effective clock rate is only about 20 Hz. Instead, by adding a rhythm, we can interleave many calculations (in this particular case about the possibility of different time warps) while the basic inputs are changing very little. Using an oscillation frequency of 50 Hz and a resolving time of 1 ms in the speech example we describe increases the effective clock rate by more than a factor of 10 compared to the effective clock rate of the UHT computation.

We believe that "time as its own representation" is a loser for processing information when the computation desired is complex but the data is slowly changing. No computer scientist would use a computer with a 24 Hz clock to analyze a movie because the movie is viewed at 24 frames a second. Biology will surely have found its way out of this "paced by the environment" dilemma. Finally, because problems are easy or hard according to how algorithms fit on hardware and according to the representation of information, the differences in operation between the system we have described and conventional ANN suggest the utility of thinking about other problems in a timing representation.

## Acknowledgements

The authors thank Sanjoy Mahajan and Erik Winfree for comments and help with preparation of the manuscript. This work was supported in part by the Center for Neuromorphic Systems Engineering as a part of the National Science Foundation Engineering Research Center Program under grant EEC-9402726. Roweis is supported by the Natural Sciences and Engineering Research Council of Canada under an NSERC 1967 Award.

## Footnotes

[1]The analog match problem of olfaction is actually viewed through olfactory receptor cells. Studies of vertebrate sensory cells have shown that each molecular species stimulates many different sensory cells, and each cell is excited by many different molecular species. The pattern of relative excitation across the population of sensory cell classes determines the odor quality in the generalist olfactory system. There are about 1000 broadly responsive cell types; thus, the olfactory systems of higher animals apparently solve an analog match problem of the type described by (2), except that the indices refer to cell types, and the actual dimension is no more than 1000.

## References

[Hopfield, 1995] Hopfield, J. (1995). Pattern recognition computation using action potential timing for stimulus representation. *Nature*, 376:33–36.

[Hopfield, 1996] Hopfield, J. (1996). Transforming neural computations and representing time. *Proceedings of the National Academy of Sciences*, 93:15440–15444.

[Hopfield and Tank, 1987] Hopfield, J. and Tank, D. (1987). Neural computation by concentrating information in time. *Proceedings of the National Academy of Sciences*, 84:1896–1900.

[Unnikrishnan et al., 1991] Unnikrishnan, K., Hopfield, J., and Tank, D. (1991). Connected digit speaker-dependent speech recognition using a neural network with time-delayed connections. *IEEE Transactions on Signal Processing*, 39:698–713.

[Unnikrishnan et al., 1992] Unnikrishnan, K., Hopfield, J., and Tank, D. (1992). Speaker-independent digit recognition using a neural network with time-delayed connections. *Neural Computation*, 4:108–119.